# A SNoW-Based Face Detector

**Ming-Hsuan Yang**     **Dan Roth**     **Narendra Ahuja**
Department of Computer Science and the Beckman Institute
University of Illinois at Urbana-Champaign
Urbana, IL 61801
mhyang@vision.ai.uiuc.edu   danr@cs.uiuc.edu   ahuja@vision.ai.uiuc.edu

## Abstract

A novel learning approach for human face detection using a network of linear units is presented. The SNoW learning architecture is a sparse network of linear functions over a pre-defined or incrementally learned feature space and is specifically tailored for learning in the presence of a very large number of features. A wide range of face images in different poses, with different expressions and under different lighting conditions are used as a training set to capture the variations of human faces. Experimental results on commonly used benchmark data sets of a wide range of face images show that the SNoW-based approach outperforms methods that use neural networks, Bayesian methods, support vector machines and others. Furthermore, learning and evaluation using the SNoW-based method are significantly more efficient than with other methods.

## 1 Introduction

Growing interest in intelligent human computer interactions has motivated a recent surge in research on problems such as face tracking, pose estimation, face expression and gesture recognition. Most methods, however, assume human faces in their input images have been detected and localized.

Given a single image or a sequence of images, the goal of face detection is to identify and locate human faces regardless of their positions, scales, orientations, poses and illumination. To support automated solutions for the above applications, this has to be done efficiently and robustly. The challenge in building an efficient and robust system for this problem stems from the fact that human faces are highly non-rigid objects with a high degree of variability in size, shape, color and texture.

Numerous intensity-based methods have been proposed recently to detect human faces in a single image or a sequence of images. Sung and Poggio [24] report an example-based learning approach for locating vertical frontal views of human faces. They use a number of Gaussian clusters to model the distributions of face and non-face patterns. A small window is moved over an image to determine whether a face exists using the estimated distributions. In [16], a detection algorithm is proposed that combines template matching and feature-based detection method using hierarchical Markov random fields (MRF) and maximum *a posteriori* probability (MAP) estimation. Colmenarez and Huang [4] apply Kullback relative information for maximal discrimination between positive and negative examples of faces. They use a family of discrete Markov processes to model faces and background patterns and estimate the density functions. Detection of a face is based on the likelihood

ratio computed during training. Moghaddam and Pentland [12] propose a probabilistic method that is based on density estimation in a high dimensional space using an eigenspace decomposition. In [20], Rowley et al. use an ensemble of neural networks to learn face and non-face patterns for face detection. Schneiderman et al. describe a probabilistic method based on local appearance and principal component analysis [23]. Their method gives some preliminary results on profile face detection. Finally, hidden Markov models [17], higher order statistics [17], and support vector machines (SVM) [14] have also been applied to face detection and demonstrated some success in detecting upright frontal faces under certain lighting conditions.

In this paper, we present a face detection method that uses the SNoW learning architecture [18, 3] to detect faces with different features and expressions, in different poses, and under different lighting conditions. SNoW (Sparse Network of Winnows) is a sparse network of linear functions that utilizes the Winnow update rule [10]. SNoW is specifically tailored for learning in domains in which the potential number of features taking part in decisions is very large, but may be unknown a priori. Some of the characteristics of this learning architecture are its sparsely connected units, the allocation of features and links in a data driven way, the decision mechanism and the utilization of an efficient update rule. SNoW has been used successfully on a variety of large scale learning tasks in the natural language domain [18, 13, 5, 19] and this is its first use in the visual processing domain.

In training the SNoW-based face detector, we use a set of 1,681 face images from Olivetti [22], UMIST [6], Harvard [7], Yale [1] and FERET [15] databases to capture the variations in face patterns. In order to compare our approach with other methods, our experiments involve two benchmark data sets [20, 24] that have been used in other works on face detection. The experimental results on these benchmark data sets (which consist of 225 images with 619 faces) show that our method outperforms all other methods evaluated on this problem, including those using neural networks [20], Kullback relative information [4], naive Bayes [23] and support vector machines [14], while being significantly more efficient computationally. Along with these experimental results we describe further experiments that provide insight into some of the theoretical and practical considerations of SNoW-based learning systems. In particular, we study the effect of learning with primitive as well as with multi-scale features, and discuss some of the sources of the success of the approach.

## 2 The SNoW System

The SNoW (Sparse Network of Winnows) learning architecture is a sparse network of linear units over a common pre-defined or incrementally learned feature space. Nodes in the input layer of the network represent simple relations over the input and are being used as the input features. Each linear unit is called a *target node* and represents relations which are of interest over the input examples; in the current application, only two target nodes are being used, one as a representation for a *face* pattern and the other for a *non-face* pattern. Given a set of relations (i.e., *types* of features) that may be of interest in the input image, each input image is mapped into a set of features which are *active* (present) in it; this representation is presented to the input layer of SNoW and propagates to the target nodes. (Features may take either binary value, just indicating the fact that the feature is active (present) or real values, reflecting its strength; in the current application, all features are binary. See Sec 3.1.) Target nodes are linked via weighted edges to (some of the) input features. Let $\mathcal{A}_t = \{i_1, \ldots, i_m\}$ be the set of features that are active in an example and are linked to the target node $t$. Then the linear unit is *active* if and only if $\sum_{i \in \mathcal{A}_t} w_i^t > \theta_t$, where $w_i^t$ is the weight on the edge connecting the $i$th feature to the target node $t$, and $\theta_t$ is its threshold.

In the current application a single SNoW *unit* which includes two subnetworks, one

for each of the targets, is used. A given example is treated autonomously by each target subnetwork; that is, an image labeled as a face is used as a positive example for the *face* target and as a negative example for the *non-face* target, and vice-versa.

The learning policy is on-line and mistake-driven; several update rules can be used within SNoW. The most successful update rule, and the only one used in this work is a variant of Littlestone's Winnow update rule, a multiplicative update rule tailored to the situation in which the set of input features is not known a priori, as in the infinite attribute model [2]. This mechanism is implemented via the sparse architecture of SNoW. That is, (1) input features are allocated in a data driven way – an input node for the feature $i$ is allocated only if the feature $i$ is active in the input image and (2) a link (i.e., a non-zero weight) exists between a target node $t$ and a feature $i$ if and only if $i$ has been active in an image labeled $t$. Thus, the architecture also supports augmenting the feature types at later stages or from external sources in a flexible way, an option we do not use in the current work.

The Winnow update rule has, in addition to the threshold $\theta_t$ at the target $t$, two update parameters: a *promotion* parameter $\alpha > 1$ and a *demotion* parameter $0 < \beta < 1$. These are being used to update the current representation of the target $t$ (the set of weights $w_i^t$) only when a mistake in prediction is made. Let $\mathcal{A}_t = \{i_1, \ldots, i_m\}$ be the set of active features that are linked to the target node $t$. If the algorithm predicts 0 (that is, $\sum_{i \in \mathcal{A}_t} w_i^t \leq \theta_t$) and the received label is 1, the active weights in the current example are *promoted* in a multiplicative fashion: $\forall i \in \mathcal{A}_t, w_i^t \leftarrow \alpha \cdot w_i^t$. If the algorithm predicts 1 ($\sum_{i \in \mathcal{A}_t} w_i^t > \theta_t$) and the received label is 0, the active weights in the current example are *demoted*: $\forall i \in \mathcal{A}_t, w_i^t \leftarrow \beta \cdot w_i^t$. All other weights are unchanged. The key property of the Winnow update rule is that the number of examples[1] it requires to learn a linear function grows linearly with the number of *relevant* features and only logarithmically with the total number of features. This property seems crucial in domains in which the number of potential features is vast, but a relatively small number of them is relevant (this does not mean that only a small number of them will be active, or have non-zero weights). Winnow is known to learn efficiently any linear threshold function and to be robust in the presence of various kinds of noise and in cases where no linear-threshold function can make perfect classification, and still maintain its abovementioned dependence on the number of total and relevant attributes [11, 9]. Once target subnetworks have been learned and the network is being evaluated, a winner-take-all mechanism selects the dominant active target node in the SNoW unit to produce a final prediction. In general, but not in this work, units' output may be cached and processed along with the output of other SNoW units to produce a coherent output.

## 3   Learning to detect faces

For training, we use a set of 1,681 face images (collected from Olivetti [22], UMIST [6], Harvard [7], Yale [1] and FERET [15] databases) which have wide variations in pose, facial expression and lighting condition. For negative examples we start with 8,422 non-face examples from 400 images of landscapes, trees, buildings, etc. Although it is extremely difficult to collect a representative set of non-face examples, the bootstrap method [24] is used to include more non-face examples during training. For positive examples, each face sample is manually cropped and normalized such that it is aligned vertically and its size is $20 \times 20$ pixels. To make the detection method less sensitive to scale and rotation variation, 10 face examples are generated from each original sample. The images are produced by randomly rotating the images by up to 15 degrees with scaling between 80% and 120%. This produces 16,810 face samples. Then, histogram equalization is performed that maps the

intensity values to expand the range of intensities. The same procedure is applied to input images in detection phase.

## 3.1 Primitive Features

The SNoW-based face detector makes use of Boolean features that encode the positions and intensity values of pixels. Let the pixel at $(x, y)$ of an image with width $w$ and height $h$ have intensity value $I(x, y)$ $(0 \leq I(x, y) \leq 255)$. This information is encoded as a feature whose index is $256(y * w + x) + I(x, y)$. This representation ensures that different points in the {position $\times$ intensity} space are mapped to different features. (That is, the feature indexed $256(y * w + x) + I(x, y)$ is *active* if and only if the intensity in position $(x, y)$ is $I(x, y)$.) In our experiments, the values for $w$ and $h$ are 20 since each face sample has been normalized to an image of $20 \times 20$ pixels. Note that although the number of potential features in our representation is 102400 ($400 \times 256$), only 400 of those are active (present) in each example, and it is plausible that many features will never be active. Since the algorithm's complexity depends on the number of active features in an example, rather than the total number of features, the sparseness also ensures efficiency.

## 3.2 Multi-scale Features

Many vision problems have utilized multi-scale features to capture the structures of an object. However, extracting detailed multi-scale features using edge or region information from segmentation is a computationally expensive task. Here we use the SNoW paradigm to extract Boolean features that represent multi-scale information. This is done in a similar way to the {position $\times$ intensity} used in Sec. 3.1, only that in this case we encode, in addition to position, the mean and variance of a multi-scale pixel. The hope is that the multi-scale feature will capture information that otherwise requires many pixel-based features to represent, and thus simplify the learning problem. Uninformative multi-scale features will be quickly assigned low weights by the learning algorithm and will not degrade performance. Since each face sample is normalized to be a rectangular image of the same size, it suffices to consider rectangular sub-images with varying size from face samples, and for each generate features in terms of the means and variances of their intensity values. Empirical results show that faces can be described effectively this way.

Instead of using the absolute values of the mean and variance when encoding the features, we discretize these values into a predefined number of classes. Since the distribution of the mean values as well as the variance values is normal, the discretization is finer near the means of these distributions. The total number of values was determined empirically to be 100, out of which 80 ended up near the mean. Given that, we use the same scheme as in Sec. 3.1 to map the {position $\times$ intensity mean $\times$ intensity variance} space into the Boolean feature space. This is done separately for four different sub-image scales, of $1 \times 1$, $2 \times 2$, $4 \times 4$ to $10 \times 10$ pixels. The multi-scale feature vector consists of active features corresponding to all these scales. The number of active features in each example is therefore $400 + 100 + 25 + 4$, although the total number of features is much larger.

In recent work we have used more sophisticated conjunctive features for this purpose yielding even better results. However, the emphasis here is that with the SNoW approach, even very simplistic features support excellent performance.

## 4 Empirical Results

We tested the SNoW-based approach with both sets of features on the two sets of images collected by Rowley [20], and Sung [24]. Each image is scanned with a rectangular window to determine whether a face exists in the window or not. To detect faces of different scales, each input image is repeatedly subsampled by a factor of 1.2 and scanned through for 10 iterations. Table 1 shows the reported

experimental results of the SNoW-based face detectors and several face detection systems using the two benchmark data sets (available at http://www.cs.cmu.edu/ ~har/ faces.html). The first data set consists of 130 images with 507 frontal faces and the second data set consists of 23 images with 155 frontal faces. There are a few hand drawn faces and cartoon faces in both sets. Since some methods use intensity values as their features, systems 1-4 and 7 discard these such hand drawn and cartoon faces. Therefore, there are 125 images with 483 faces in test set 1 and 20 images with 136 faces in test set 2 respectively. The reported detection rate is computed as the ratio between the number of faces detected in the images by the system and the number of faces identified there by humans. The number of false detections is the number of non-faces detected as faces.

It is difficult to evaluate the performance of different methods even though they use the same benchmark data sets because different criteria (e.g. training time, number of training examples involved, execution time, number of scanned windows in detection) can be applied to favor one over another. Also, one can tune the parameters of one's method to increase the detection rates while increasing also the false detections. The methods using neural networks [20], distribution-based [24], Kullback relative information [4] and naive Bayes [23] report several experimental results based on different sets of parameters. Table 1 summarizes the best detection rates and corresponding false detections of these methods. Although the method in [4] has the highest detection rates in one benchmark test, this was done by significantly increasing the number of false detections. Other than that, it is evident that the SNoW-based face detectors outperforms others in terms of the overall performance. These results show the credibility of SNoW for these tasks, as well

Table 1: Experimental results on images from test set 1 (125 images with 483 faces) in [20] and test set 2 (20 images with 136 faces) in [24] (see text for details)

| Method | Test Set 1 | | Test Set 2 | |
|---|---|---|---|---|
| | Detect Rate | False Detects | Detect Rate | False Detects |
| **SNoW w/ primitive features** | **94.2%** | **84** | **93.6%** | **3** |
| **SNoW w/ multi-scale features** | **94.8%** | **78** | **94.1%** | **3** |
| Mixture of factor analyzers [26] | 92.3% | 82 | 89.4% | 3 |
| Fisher linear discriminant [27] | 93.6% | 74 | 91.5% | 1 |
| Distribution-based [24] | N/A | N/A | 81.9% | 13 |
| Neural network [20] | 92.5% | 862 | 90.3% | 42 |
| Naive Bayes [23] | 93.0% | 88 | 91.2% | 12 |
| Kullback relative information [4] | 98.0% | 12758 | N/A | N/A |
| Support vector machine [14] | N/A | N/A | 74.2% | 20 |

as exhibit the improvement achieved by increasing the expressiveness of the features. This may indicate that further elaboration of the features, which can be done in a very general and flexible way within SNoW, would yield further improvements.

In addition to comparing feature sets, we started to investigate some of the reasons for the success of SNoW in this domain, which we discuss briefly below. Two potential contributions are the Winnow update rule and the architecture. First, we studied the update rule in isolation, independent of the SNoW architecture. The results we got when using the Winnow simply as a discriminator were fairly poor (63.9%/65.3% for Test Set 1, primitive and multi-scale features, respectively, and similar results for the Test Set 2.). The results are not surprising, given that Winnow is used here only as a discriminator and is using only positive weights. Investigating the architecture in isolation reveals that weighting or discarding features based on their contribution to mistakes during training, as is done within SNoW, is crucial. Considering the active features uniformly (separately for faces and non-faces) yields poor results. Specifically, studying the resulting SNoW network shows that the total number of features that were active with non-faces is 102,208, out of 102,400 possible

(primitive) features. The total number of active features in faces was only 82,608, most of which are active only a few times. In retrospect, this is clear given the diverse set of images used as negative examples, relative to the somewhat restricted (by nature) set of images that constitute faces. (Similar phenomenon occurs with the multi-scale features, where the numbers are 121572 and 90528, respectively, out of 135424.) Overall it exhibits that the architecture, the learning regime and the update rule all contribute significantly to the success of the approach.

Figure 1 shows some faces detected in our experiments. Note that profile faces and faces under heavy illumination are detected. Experimental results show that profile faces and faces under different illumination are detected very well by our method. Note that although there may exist several detected faces around each face, only one window is drawn to enclose each detected face for clear presentation.

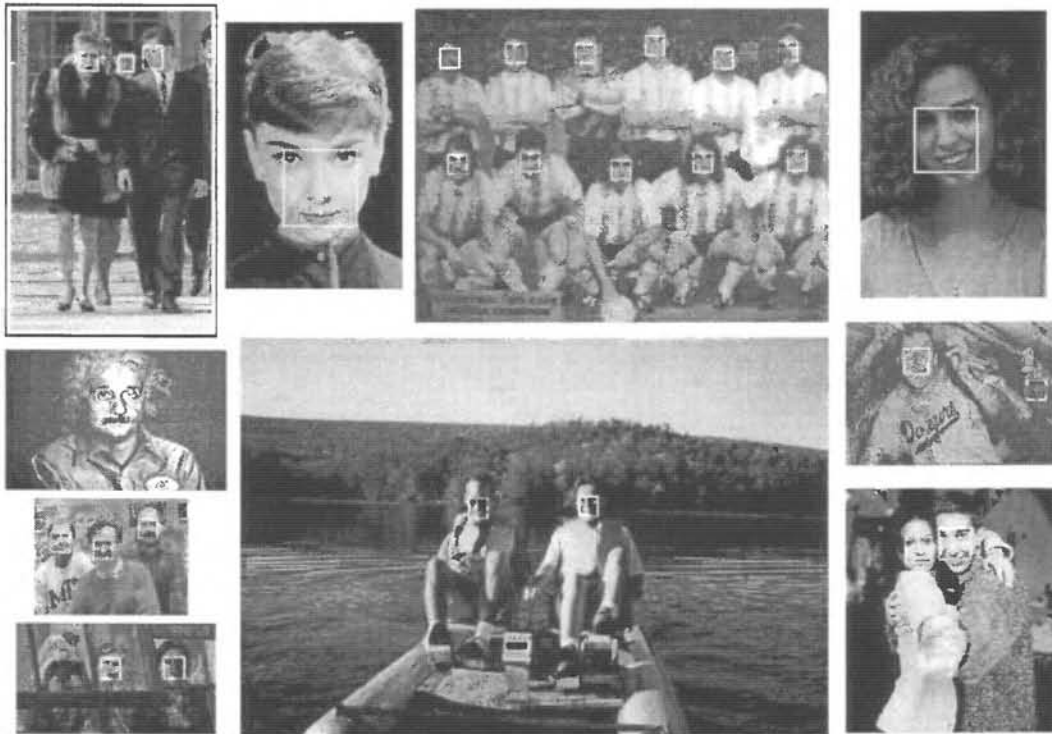

Figure 1: Sample experimental results using our method on images from two benchmark data sets. Every detected face is shown with an enclosing window.

## 5  Discussion and Conclusion

Many theoretical and experimental issues are to be addressed before a learning system of this sort can be used to detect faces efficiently and robustly under general conditions. In terms of the face detection problem, the presented method is still not able to detect rotated faces. A recent method [21], addresses this problem by building upon a upright face detector [20] and rotating each test sample to upright position. However, it suffers from degraded detection rates and more false detections. Given our results, we believe that the SNoW approach, if adapted in similar ways, would generalize very well to detect faces under more general conditions.

In terms of the SNoW architecture, although the main ingredients of it are understood theoretically, more work is required to better understand its strengths. This is increasingly interesting given that the architecture has been found to perform very well in large-scale problem in the natural language domain as well

The contributions of this paper can be summarized as follows. We have introduced the SNoW learning architecture to the domain of visual processing and described an approach that detect faces regardless of their poses, facial features and illumination conditions. Experimental results show that this method outperforms other methods in terms of detection rates and false detectionss, while being more efficient both in learning and evaluation.

## Footnotes

[1]In the on-line setting [10] this is usually phrased in terms of a mistake-bound but is known to imply convergence in the PAC sense [25, 8].

# References

[1] P. Belhumeur, J. Hespanha, and D. Kriegman. Eigenfaces vs. fisherfaces: Recognition using class specific linear projection. *IEEE Transactions on Pattern Analysis and Machine Intelligence*, 19(7):711–720, 1997.

[2] A. Blum. Learning boolean functions in an infinite attribute space. *Machine Learning*, 9(4):373–386, 1992.

[3] A. Carleson, C. Cumby, J. Rosen, and D. Roth. The SNoW learning architecture. Technical Report UIUCDCS-R-99-2101, UIUC Computer Science Department, May 1999.

[4] A. J. Colmenarez and T. S. Huang. Face detection with information-based maximum discrimination. In *Proceedings of the IEEE Computer Society Conference on Computer Vision and Pattern Recognition*, pages 782–787, 1997.

[5] A. R. Golding and D. Roth. A winnow based approach to context-sensitive spelling correction. *Machine Learning*, 34:107–130, 1999. Special Issue on Machine Learning and Natural Language.

[6] D. B. Graham and N. M. Allinson. Characterizing virtual eigensignatures for general purpose face recognition. In H. Wechsler, P. J. Phillips, V. Bruce, F. Fogelman-Soulie, and T. S. Huang, editors, *Face Recognition: From Theory to Applications*, volume 163 of *NATO ASI Series F, Computer and Systems Sciences*, pages 446–456. Springer, 1998.

[7] P. Hallinan. *A Deformable Model for Face Recognition Under Arbitrary Lighting Conditions*. PhD thesis, Harvard University, 1995.

[8] D. Helmbold and M. K. Warmuth. On weak learning. *Journal of Computer and System Sciences*, 50(3):551–573, June 1995.

[9] J. Kivinen and M. K. Warmuth. Exponentiated gradient versus gradient descent for linear predictors. In *Proceedings of the Annual ACM Symposium on the Theory of Computing*, 1995.

[10] N. Littlestone. Learning quickly when irrelevant attributes abound: A new linear-threshold algorithm. *Machine Learning*, 2:285–318, 1988.

[11] N. Littlestone. Redundant noisy attributes, attribute errors, and linear threshold learning using winnow. In *Proceedings of the fourth Annual Workshop on Computational Learning Theory*, pages 147–156, 1991.

[12] B. Moghaddam and A. Pentland. Probabilistic visual learning for object recognition. *IEEE Transactions on Pattern Analysis and Machine Intelligence*, 19(7):696–710, 1997.

[13] M. Munoz, V. Punyakanok, D. Roth, and D. Zimak. A learning approach to shallow parsing. In *EMNLP-VLC'99, the Joint SIGDAT Conference on Empirical Methods in Natural Language Processing and Very Large Corpora*, June 1999.

[14] E. Osuna, R. Freund, and F. Girosi. Training support vector machines: an application to face detection. In *Proceedings of the IEEE Computer Society Conference on Computer Vision and Pattern Recognition*, pages 130–136, 1997.

[15] P. J. Phillips, H. Moon, S. Rizvi, and P. Rauss. The feret evaluation. In H. Wechsler, P. J. Phillips, V. Bruce, F. Fogelman-Soulie, and T. S. Huang, editors, *Face Recognition: From Theory to Applications*, volume 163 of *NATO ASI Series F, Computer and Systems Sciences*, pages 244–261. Springer, 1998.

[16] R. J. Qian and T. S. Huang. Object detection using hierarchical mrf and map estimation. In *Proceedings of the IEEE Computer Society Conference on Computer Vision and Pattern Recognition*, pages 186–192, 1997.

[17] A. N. Rajagopalan, K. S. Kumar, J. Karlekar, R. Manivasakan, and M. M. Patil. Finding faces in photographs. In *Proceedings of the Sixth International Conference on Computer Vision*, pages 640–645, 1998.

[18] D. Roth. Learning to resolve natural language ambiguities: A unified approach. In *Proceedings of the Fifteenth National Conference on Artificial Intelligence*, pages 806–813, 1998.

[19] D. Roth and D. Zelenko. Part of speech tagging using a network of linear separators. In *COLING-ACL 98, The 17th Int. Conference on Computational Linguistics*, pages 1136–1142, 1998.

[20] H. Rowley, S. Baluja, and T. Kanade. Neural network-based face detection. *IEEE Transactions on Pattern Analysis and Machine Intelligence*, 20(1):23–38, 1998.

[21] H. Rowley, S. Baluja, and T. Kanade. Rotation invariant neural network-based face detection. In *Proceedings of the IEEE Computer Society Conference on Computer Vision and Pattern Recognition*, pages 38–44, 1998.

[22] F. S. Samaria. *Face Recognition Using Hidden Markov Models*. PhD thesis, University of Cambridge, 1994.

[23] H. Schneiderman and T. Kanade. Probabilistic modeling of local appearance and spatial relationships for object recognition. In *Proceedings of the IEEE Computer Society Conference on Computer Vision and Pattern Recognition*, pages 45–51, 1998.

[24] K.-K. Sung and T. Poggio. Example-based learning for view-based human face detection. *IEEE Transactions on Pattern Analysis and Machine Intelligence*, 20(1):39–51, 1998.

[25] L. G. Valiant. A theory of the learnable. *Commun. ACM*, 27(11):1134–1142, Nov. 1984.

[26] M.-H. Yang, N. Ahuja, and D. Kriegman. Face detection using a mixture of factor analyzers. In *Proceedings of the IEEE International Conference on Image Processing*, 1999.

[27] M.-H. Yang, N. Ahuja, and D. Kriegman. Mixtures of linear subspaces for face detection. In *Proceedings of the Foruth IEEE International Conference on Automatic Face and Gesture Recognition*, 2000.